# Learning with Multiple Labels

**Rong Jin***
*School of Computer Science
Carnegie Mellon University
Pittsburgh, PA 15213, USA
*rong@cs.cmu.edu*

**Zoubin Ghahramani†***
†Gatsby Computational Neuroscience Unit
University College London
London WC1N 3AR, UK
*zoubin@gatsby.ucl.ac.uk*

## Abstract

In this paper, we study a special kind of learning problem in which each training instance is given a set of (or distribution over) candidate class labels and only one of the candidate labels is the correct one. Such a problem can occur, e.g., in an information retrieval setting where a set of words is associated with an image, or if classes labels are organized hierarchically. We propose a novel discriminative approach for handling the ambiguity of class labels in the training examples. The experiments with the proposed approach over five different UCI datasets show that our approach is able to find the correct label among the set of candidate labels and actually achieve performance close to the case when each training instance is given a single correct label. In contrast, naïve methods degrade rapidly as more ambiguity is introduced into the labels.

## 1 Introduction

Supervised and unsupervised learning problems have been extensively studied in the machine learning literature. In supervised classification each training instance is associated with a single class label, while in unsupervised classification (i.e. clustering) the class labels are not known. There has recently been a great deal of interest in partially- or semi-supervised learning problems, where the training data is a mixture of both labeled and unlabelled cases. Here we study a new type of semi-supervised learning problem.

We generalize the notion of supervision by thinking of learning problems where multiple candidate class labels are associated with each training instance, and it is assumed that only one of the candidates is the correct label. For a supervised classification problem, the set of candidate class labels for every training instance contains only one label, while for an unsupervised learning problem, the set of candidate class labels for each training instance counts in all the possible class labels. For a learning problem with the mixture of labeled and unlabelled training data, the number of candidate class labels for every training instance can be either one or the total number of different classes.

Here we study the general setup, i.e. a learning problem when each training instance is assigned to a subset of all the class labels (later, we further generalize this to

include arbitrary distributions over the class labels). For example, there may be 10 different classes and each training instance is given two candidate class labels and one of the two given labels is correct. This learning problem is more difficult than supervised classification because for each training example we don't know which class among the given set of candidate classes is actually the target. For easy reference, we called this class of learning problems 'multiple-label' problems.

In practice, many real problems can be formalized as a 'multiple-label' problem. For example, the problem of having several different class labels for a single training example can be caused by the disagreement between several assessors.[1] Consider the scenario when two assessors are hired to label the training data and sometimes the two assessors give different class labels to the same training example. In this case, we will have two class labels for a single training instance and don't know which, if any, is actually correct. Another scenario that can cause multiple class labels to be assigned to a single training example is when there is a hierarchical structure over the class labels and some of the training data are given the labels of the internal nodes in the hierarchy (i.e. superclasses) instead of the labels of the leaf nodes (subclasses). Such hierarchies occur, for example, in bioinformatics where proteins are regularly classified into superfamilies and families. For such hierarchical labels, we can treat the label of internal nodes as a set of the labels on the leaf nodes.

## 2 Related Work

First of all, we need to distinguish this 'multiple-label' problem from the problem where the classes are not mutually exclusive and therefore each training example is allowed several class labels [4]. There, even though each training example can have multiple class labels, all the assigned class labels are actually correct labels while in 'multiple-label' problems only one of the assigned multiple labels is the target label for the training instance.

The essential difficulty of 'multiple-label' problems comes from the ambiguity in the class labels for training data, i.e. among the several labels assigned to every training instance only one is presumed to be the correct one and unfortunately we are not informed which one is the target label. A similar difficulty appears in the problem of classification from labeled and unlabeled training data. The difference between the 'multiple-label' problem and the labeled/unlabeled classification problem is that in the former only a subset of the class labels can be the candidate for the target label, while in the latter any class label can be the candidate. As will be shown later, this constraint makes it possible for us to build up a purely *discriminative* approach while for learning problems using unlabeled data people usually take a generative approach and model properties of the input distribution.

In contrast to the 'multiple-label' problem, there is a set of problems named 'multiple-instance' problems [3] where instances are organized into 'bags' of several instances, and a class label is tagged for every bag of instances. In the 'multiple-instance' problem, at least one of the instances within each bag corresponds to the label of the bag and all other instances within the bag are just noise. The difference between 'multiple-label' problems and 'multiple-instance' problems is that for 'multiple-label' problems the ambiguity lies on the side of class labels while for 'multiple-instance' problem the ambiguity comes from the instances within the bag.

The most related work to this paper is [6], where a similar problem is studied using the logistic regression method. Our framework is completely general for any discriminative model and incorporates non-uniform 'prior' on the labels.

## 3 Formal Description of the 'Multiple-label' Problem

As described in the introduction, for a 'multiple-label' problem, each training instance is associated with a set of candidate class labels, only one of which is the target label for that instance. Let $x_i$ be the input for the i-th training example, and $S_i$ be the set of candidate class labels for the i-th training example. Our goal is to find the model parameters $\theta \in \Theta$ in some class of models $\mathsf{M}$, i.e. a parameterized classifier with parameters $\theta$ which maps inputs to labels, so that the predicted class label $y$ for the i-th training example has a high probability to be a member of the set $S_i$. More formally, using the maximum likelihood criterion and the assumption of i.i.d. assignments, this goal can be simply stated as

$$\theta* = \arg\max_{\theta} \prod_i p(y \in S_i \mid x_i, \theta) = \arg\max_{\theta} \sum_i \log \sum_{y \in S_i} p(y \mid x_i, \theta) \quad (1)$$

## 4 Description of the Discriminative Model for the 'Multiple-label' Problem

Before discussing the discriminative model for the 'multiple-label' problem, let's look at the standard discriminative model for supervised classification. Let $\hat{p}(y \mid x_i)$ stand for some given conditional distribution of class labels for the training instance $x_i$ and $p(y \mid x_i, \theta)$ be the model-based conditional distribution for the training data $x_i$ to have the class label $y$. A common and sensible criterion for finding model parameters $\theta^*$ is to minimize the KL divergence between the given conditional distributions and the model-based distributions, i.e.

$$\theta* = \arg\min_{\theta} \left\{ \sum_i \sum_y \hat{p}(y \mid x_i) \log \frac{\hat{p}(y \mid x_i)}{p(y \mid x_i, \theta)} \right\} \quad (2)$$

For supervised learning problems, the class label for every training instance is known. Therefore, the given conditional distribution of the class label for every training instance is a delta function or $\hat{p}(y \mid x_i) = \delta(y, y_i)$ where $y_i$ is the given class label for the i-th instance. With this, it can be easily shown that Eqn. (2) will be simplified as maximum likelihood criterion. For the 'multiple-label' problem, each training instance $x_i$ is assigned to a set of candidate class labels $S_i$ and therefore Eqn. (2) can be rewritten as:

$$\theta* = \arg\min_{\theta} \left\{ \sum_i \sum_{y \in S_i} \hat{p}(y \mid x_i) \log \frac{\hat{p}(y \mid x_i)}{p(y \mid x_i, \theta)} \right\} \quad (3)$$

with the constraints $\forall i \; \sum_{y \in S_i} \hat{p}(y \mid x_i) = 1$. $\quad (4)$

In the 'multiple-label' problem the distribution of class labels $\hat{p}(y \mid x_i)$ is unknown except for the constraint that the target class label for every training example is a member of the corresponding set of candidate class labels. A simple solution to the problem of unknown label distribution is to assume it is uniform, i.e. $\hat{p}(y \mid x_i) = \hat{p}(y' \mid x_i)$ for any $y, y' \in S_i$. Then, Eqn. (3) can be simplified to:

$$\theta^* = \arg\min_{\theta} \left\{ \sum_i \frac{1}{|S_i|} \sum_{y \in S_i} \log\left(\frac{1}{|S_i| \, p(y \mid x_i, \theta)}\right) \right\} = \arg\max_{\theta} \left\{ \sum_i \frac{1}{|S_i|} \sum_{y \in S_i} \log p(y \mid x_i, \theta) \right\}, \qquad (5)$$

which corresponds to minimizing the KL divergence (2) to a uniform over $S_i$. For the case of multiple assessors giving differing labels to the data, discussed in the introduction, this corresponds to concatenating the labeled data sets. Standard learning algorithms can be applied to learn the conditional model $p(y \mid x, \theta)$. For later reference, we called this simple idea the 'Naïve Model'.

A better solution than the 'Naïve Model' is to disambiguate the label association, i.e. to find which label among the given set is more appropriate than the others and use the appropriate label for training. It turns out that it is possible to apply the EM algorithm [2] to accomplish this goal, resulting in a procedure which iterates between disambiguating and classifying. Starting with the assumption that every class label within the set is equally likely, we train a conditional model $p(y \mid x, \theta)$. Then, with the help of this conditional model, we estimate the label distribution $\hat{p}(y \mid x_i)$ for each data point. With these label distributions, we refit the conditional model $p(y \mid x, \theta)$ and so on. More formally, this idea can be expressed as follows:

First, we estimate the conditional model based on the assumed or estimated label distribution according to Eqn. (3). This step corresponds to the M-step in the EM algorithm. Then, in the E-step, new label distributions are estimated by maximizing Eqn. (3) w.r.t. $\hat{p}(y \mid x_i)$ under the constraints (4), resulting in:

$$\hat{p}(y \mid x_i) = \begin{cases} \dfrac{p(y \mid x_i, \theta)}{\sum_{y' \in S_i} p(y' \mid x_i, \theta)} & \forall y \in S_i \\ 0 & \text{otherwise} \end{cases} \qquad (6)$$

Importantly, this procedure optimizes the objective function in Eqn. (1), by the usual EM proof. The negative of the KL divergence in Eqn. (3) is a lower bound on the log likelihood (1) by Jensen's inequality. Substituting Eqn. (6) for $\hat{p}(y \mid x_i)$ into (3) we obtain equality. For easy reference, we called this model the 'EM Model'.

In some 'multiple-label' problems, information on which class label within the set $S_i$ is more likely to be the correct one can be obtained. For example, if three assessors manually label the training data, in some cases two assessors will agree on the class label and the other doesn't. We should give more weights to the labels that are agreed by two assessors and low weights to the labels that are chosen by only one. To accommodate prior information on the class labels, we generalize the previous framework so that the estimated label distribution $\hat{p}(y \mid x_i)$ has low relative entropy with the prior on the class labels. Therefore, the objective function (1) and its EM-bound (4) can be modified to be

$$\theta^* = \arg\min_{\theta} \left\{ \sum_i \sum_{y \in S_i} \hat{p}(y \mid x_i) \log \frac{\hat{p}(y \mid x_i)}{\pi_{i,y}} - \sum_i \sum_{y \in S_i} \hat{p}(y \mid x_i) \log p(y \mid x_i, \theta) \right\} \qquad (7)$$

where $\pi_{i,y}$ is the prior probability for the i-th training example to have class label $y$. The first term in the objective function (7) encourages the estimated label distribution to be consistent with the prior distribution of class labels and the second term encourages the prediction of the model to be consistent with the estimated label distribution. The objective (7) is an upper bound on $-\sum_i \log \sum_{y \in S_i} \pi_{i,y} p(y \mid x_i, \theta)$.

When there is no prior information about which class label within the given set is preferable we can set $\pi_{i,y} = 1/|S_i|$ and Eqn. (7) becomes

$$
\begin{aligned}
\theta^* &= \arg\min_\theta \left\{ \sum_i \sum_{y \in S_i} \hat{p}(y|x_i) \log \frac{\hat{p}(y|x_i)}{1/|S_i|} - \sum_i \sum_{y \in S_i} \hat{p}(y|x_i) \log p(y|x_i,\theta) \right\} \\
&= \arg\min_\theta \left\{ \sum_i \sum_{y \in S_i} \hat{p}(y|x_i) \log \frac{\hat{p}(y|x_i)}{p(y|x_i,\theta)} + \sum_i \log|S_i| \right\} = \arg\min_\theta \left\{ \sum_i \sum_{y \in S_i} \hat{p}(y|x_i) \log \frac{\hat{p}(y|x_i)}{p(y|x_i,\theta)} \right\}
\end{aligned}
\tag{7'}
$$

Eqn. (7') is identical to Eqn. (3), which shows that when there is no prior knowledge on the class label distribution, we revert back to the 'EM Model'.

Again we can optimize Eqn. (7) using the EM algorithm, estimating the label distribution $\hat{p}(y|x_i)$ in the E step fitting any standard discriminative model for $p(y|x,\theta)$ in the M step. The label distribution that optimizes (7) in the E step is: $p(y|x_i) = \pi_{i,y} p(y|x_i,\theta)/\sum_{y' \in S_i} \pi_{i,y'} p(y'|x_i,\theta)$, and 0 otherwise. As we would expect, the label distribution $\hat{p}(y|x_i)$ trades off both the prior $\pi_{i,y}$ and the model-based prediction $p(y|x_i,\theta)$. We will call this model '**EM+Prior Model**'.

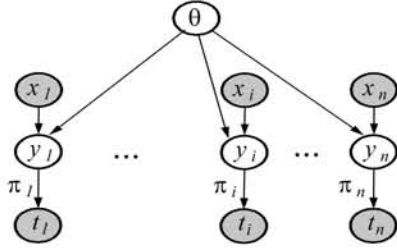

**Figure 1**: Diagram for graphic model interpretation of 'EM+Prior' model

The 'EM+Prior Model' can also be interpreted from the viewpoint of a graphical model. The basic idea is illustrated in Figure 1, where the random variable $t_i$ represents the event that the true label $y_i$ belongs to the label set $S_i$. For the 'EM+Prior' model, $\pi_{i,y}$ actually plays the role of a likelihood or noise model where, where $p(y \in S_i | x_i, \theta)$ in Eqn. (1) is replaced as in Eqn. (8). From this point of view, generalizing to Bayesian learning and regression is easy.

$$
p(t_i = 1|x_i,\theta) = \sum_{y \in S_i} p(t_i = 1|y) p(y|x_i,\theta) = \sum_{y \in S_i} \pi_{i,y} p(y|x_i,\theta)
\tag{8}
$$

## 5   Experiments

The goal of our experiments is to answer the following questions:

1. *Is the 'EM Model' better than the 'Naïve Model'?* The difference between the 'EM Model ' and the 'Naïve Model' for the 'multiple-label' problems is that the 'Naïve Model' makes no effort in finding the correct label within the given label set while the 'EM Model' applies the EM algorithm to clarify the ambiguity in the class label. Therefore, in this experiment, we need to justify empirically whether the effort in disambiguating class labels is effective.

2. *Will prior knowledge help the model?* The difference between the 'EM Model' and the 'EM+Prior Model' is that the 'EM+Prior Model' takes advantage of prior knowledge on the distribution of class labels for instances. However, since sometimes the prior knowledge on the class label can be misleading, we need to test the robustness of the 'EM+Prior Model' to such noisy prior knowledge.

### 5.1   Experimental Data

Since there don't exist standard data sets with training instances assigned to multiple class labels, we actually create several data sets with multiple class labels

from the UCI classification datasets. To make our experiments more realistic, we tried two different methods of creating datasets with multiple class labels:

• *Random Distractors.* For every training instance, in addition to the original assigned label, several randomly selected labels are added to the label candidate set. We varied the number of added classes to test reliability of our algorithm.

• *Naïve Bayes Distractors.* In the previous method, the added class labels are randomly selected and therefore independent from the original class label. However, we usually expect that distractors are in the candidate set should be correlated with the original label. To simulate this realistic situation, we use the output of a Naïve Bayes (NB) classifier as an additional member of the class label candidate set.[1] First, a Naïve Bayes classifier using Gaussian generation models is trained on the dataset. Then, the trained NB classifier is asked to predict the class label of the training data. When the output of the NB classifier differs from the original label, it is added as a candidate label. Otherwise, a randomly selected label is added to the candidate set. Since the NB classifier errors are not completely random, they should have some correlation with the originally assigned labels.

In these experiments we chose a simple maximum entropy (ME) model [5] as the basic discriminative model, which expresses a conditional probability $p(y\,|\,\vec{x},\vec{\theta})$ in an exponential form, i.e. $p(y\,|\,\vec{x},\vec{\theta}) = \exp(\vec{\theta}\cdot\vec{x})/Z(\vec{x})$ where $\vec{x}$ is the input feature vector and $Z(\vec{x})$ is the normalization constant which ensures that the conditional probabilities over all different classes $y$ sum to 1.

**Table 1**: Information about five UCI datasets that are used in the experiments

| Class Name | ecoli | wine | pendigit | iris | glass |
|---|---|---|---|---|---|
| Number of Instances | 327 | 178 | 2000 | 154 | 204 |
| Number of Classes | 5 | 3 | 10 | 3 | 5 |
| Number of Features | 7 | 13 | 16 | 14 | 10 |
| % NB_Output≠Assigned_Label | 15% | 8% | 22.3% | 13.3% | 16.6% |
| Error Rate for ME on clean data (10-fold cross validation) | 12.6% | 3.7% | 9% | 5.7% | 9.7% |

Five different UCI datasets were selected as the testbed for experiments. Information about these datasets is listed in Table 1. For each dataset, the 10-fold cross validation results for the ME model together with the percentage of time the NB output differs from the originally assigned label are also listed in Table 1.

## 5.2 Experiment Results (I): 'Naïve Model' vs. 'EM Model'

Table 2 lists the results for the 'Naïve Model' and 'EM Model' over a varied number of additional class labels created by the 'random distractor' and the 'Naïve Bayes' distractor. Since 'wine' and 'iris' datasets only have 3 different classes, the maximum additional class labels for these two datasets is 1. Therefore, there is no experiment result for the case of 2 or 3 distractor class labels for 'wine' and 'iris'.

As shown in Table 2, for the random distractor, the 'EM Model' substantially outperforms the 'Naïve Model' in all cases. Particularly, for the 'wine' and 'iris' datasets, by introducing an additional class label to every training instance, there is only one class label left out of the class label candidates and yet the performance of the 'EM Model' is still close to the case when there are no additional class labels.

Meanwhile, the 'Naïve Model' degrades significantly for both cases, i.e. from 3.7% to 10.0% for 'wine' and 5.7% to 18.5% for 'iris'. Therefore, we can conclude that the 'EM Model' is able to reduce the noise caused by randomly added class labels.

**Table 2**: Average 10-fold cross validation error rates for both 'Naïve Model' and 'EM Model'

| Class Name | | ecoli | wine | pendigit | iris | glass |
|---|---|---|---|---|---|---|
| 1 extra label by random distracter | Naive | 17.3% | 10% | 14.2% | 18.5% | 24.9% |
| | EM | 13.6% | 4.4% | 8.9% | 5.2% | 12.9% |
| 2 extra labels by random distracter | Naive | 20.7% | | 15.4% | | 44.9% |
| | EM | 14.9% | | 9.4% | | 12% |
| 3 extra labels by random distracter | Naive | 25.8% | | 17.6% | | 34.6% |
| | EM | 18.3% | | 11.7% | | 33.5% |
| 1 extra label by NB distracter | Naive | 22.4% | 15.7% | 17.2% | 18.5% | 27.7% |
| | EM | 14.6% | 6.8% | 15.4% | 6.7% | 20.6% |

Secondly, we compare the performance of these two models over a more realistic setup for the 'multiple-label' problem where the distractor identity is correlated with the true label (simulated by using the NB distractor). Table 1 gives the percentage of times when the trained Naïve Bayes classifier disagreed with the 'true' labels, which is also the percentage of the additional class labels that is created by the 'Naïve Bayes distracter'. The last row of Table 2 shows the performance of these two models when the additional class labels are introduced by the 'NB distracter'. Again, the 'EM Model' is significantly better than 'Naïve Model'. For dataset 'ecoli', 'wine' and 'iris', the averaged error rates of the 'EM Model' are very close to the cases when there are no distractor class labels. Therefore, we can conclude that the 'EM Model' is able to reduce the noise caused not only by random label ambiguity but also by some systematic label ambiguity.

## 5.3 Experiment Results (II): 'EM Model' vs. 'EM+Prior Model'

**Table 3**: Average 10-fold cross validation error rates for 'EM+Prior Model' over five UCI datasets.

| Class Name | | ecoli | wine | pendigit | iris | glass |
|---|---|---|---|---|---|---|
| 1 extra label by random distracter | Perfect | 13.3% | 3.7% | 8.7% | 5.2% | 12.4% |
| | Noisy | 13.3% | 3.2% | 9.0% | 18.5% | 12.9% |
| 2 extra labels by random distracter | Perfect | 13.6% | | 9.0% | | 12.5% |
| | Noisy | 13.9% | | 9.4% | | 13.6% |
| 3 extra labels by random distracter | Perfect | 12.6% | | 10.0% | | 12.4% |
| | Noisy | 13.9% | | 11.0% | | 16.8% |
| 1 extra label by NB distracter | Perfect | 13.9% | 5.0% | 13.4% | 5.2% | 16.7% |
| | Noisy | 15.3% | 6.2% | 14.2% | 6.7% | 19.0% |

In this subsection, we focus on whether the information from a prior distribution on class labels can improve the performance. In this experiment, we study two cases:

- *'Perfect Case'*. Here the guidance of the prior distribution on class labels is always correct. In our experiments for every training instance $x_i$ we set the probability $\pi_{i,y_i}$ twice as large for the correct $y_i$ as for other $\pi_{i,y \neq y_i}$.

- *'Noisy Case'*. For this case, we only allow the guidance of the prior distribution on the class label to be correct 70% of the time. With this setup, we are able to see if the 'EM+Prior Model' is robust to noise in the prior distribution.

Table 3 lists the results for 'EM+Prior Model' under both 'Perfect' and 'Noisy' situations over five different collections. In the 'perfect case', the averaged error rates of 'EM+Prior Model' are quite close to the case when there is no label ambiguity at all (see Table 1). Moreover, the performance of the 'Noisy case' is also close to that of the 'Perfect case' for most data sets listed in Table 3. Therefore, we can conclude that our 'EM+Prior Model' is able to take advantage of the prior distribution on class labels even when some of the 'guidance' is not correct.

## 6   Conclusions and Future Work

We introduced the 'multiple-label' problem and proposed a discriminative framework that is able to clarify the ambiguity between labels. Although it is discriminative, this framework is firmly grounded in the EM algorithm for maximum likelihood estimation. The framework was generalized to take advantage of prior knowledge on which class label is more likely to be the target label. Our experiments clearly indicate that the proposed discriminative model is robust to the addition of noisy class labels and to errors in the prior distribution over class labels.

The idea of this framework, allowing the target distribution $\hat{p}(y \mid x_i)$ to be inferred from the classifier itself, can be extended in many different ways. We outline several promising directions which we hope to explore. (1) It should be possible to extend this framework to **function approximation**, where $y \in \Re$, and ranges or distributions are given for the target. In this case, it may be useful to parameterize $\hat{p}(y \mid x_i)$ to simplify the resulting variational optimization problem. (2) We have focused on maximum likelihood; however **Bayesian** generalizations, where the goal is to compute a posterior distribution over $\theta$ given ambiguously labeled data would be interesting. (3) It is possible to use these ideas as a framework for **combining multiple models**. Each model is trained on a small labeled data set and predicts labels on a large unlabeled data set. These predicted labels can be combined with the small set to form a larger *multiply-labeled* data set (since not all models will agree). This larger data set can be used to train a more complex model. (4) It is possible to extend this framework to handle the presence of label noise and to combine it with the multiple-instance problem [3].

## Footnotes

[1] Observer disagreement has been modeled using the EM algorithm [1]. Our multiple-label framework differs in that we don't know which observer assigned which label to each case. This would be an interesting direction to extend our framework.

[1] Naïve Bayes distractor should not be confused with the multiple-label Naïve Model.

## References

[1] A. P. Dawid and A. M. Skene (1979) Maximum likelihood estimation of observer error-rates using the *EM* algorithm. *Applied Statistics* **28**:20-28.

[2] A. Dempster, N. Laird and D. Rubin (1977), Maximum likelihood from incomplete data via the EM algorithm, *Journal of the Royal Statistical Society*, 39 (Series B), 1-38.

[3] T. G. Dietterich, R. H. Lathrop, and T. L.-Perez (1997) Solving the multiple-instance problem with axis-parallel rectangles, *Artificial Intelligence*, 89(1-2), pp. 31-71.

[4] A. McCallum (1999) Multi-label text classification with a mixture model trained by EM, *AAAI'99 Workshop on Text Learning*.

[5] S. Della Pietra, V. Della Pietra and J. Lafferty (1997) Inducing features of random fields, *IEEE Transactions on Pattern Analysis and Machine Intelligence*, 19(4): 380-393.

[6] Y. Grandvalet (2002), Logistic regression for partial labels, *9th Information Processing and Management of Uncertainty in Knowledge-based System* (*IPMU'02*), pp. 1935-1941.
